# A Micropower CMOS Adaptive Amplitude and Shift Invariant Vector Quantiser

**Richard J. Coggins, Raymond J.W. Wang and Marwan A. Jabri**
Computer Engineering Laboratory
School of Electrical and Information Engineering, J03
University of Sydney, 2006, Australia.
{richardc, jwwang, marwan}@sedal.usyd.edu.au

## Abstract

In this paper we describe the architecture, implementation and experimental results for an Intracardiac Electrogram (ICEG) classification and compression chip. The chip processes and vector-quantises 30 dimensional analogue vectors while consuming a maximum of 2.5 $\mu$W power for a heart rate of 60 beats per minute (1 vector per second) from a 3.3 V supply. This represents a significant advance on previous work which achieved ultra low power supervised morphology classification since the template matching scheme used in this chip enables unsupervised blind classification of abnormal rhythms and the computational support for low bit rate data compression. The adaptive template matching scheme used is tolerant to amplitude variations, and inter- and intra-sample time shifts.

## 1  INTRODUCTION

Implantable cardioverter defibrillators (ICDs) are devices used to monitor the electrical activity of the heart muscle and to apply appropriate levels of electrical stimulation if abnormal conditions are detected. Despite the considerable success of ICDs they suffer from a number of limitations including an inability to detect and treat some abnormal heart rhythms and limited data recording capabilities.

We have previously shown that micropower analogue Multi-Layer Perceptron (MLP) neural networks can be trained to separate such arrhythmia [4]. However, MLPs are best suited to learning the boundary between classes whereas a vector quantization scheme allows a measure of the probability density of the morphological types to be estimated.

Many analogue vector quantiser (VQ) chips have been reported in the literature. For example, a 16×256 500 kHz 50 mW 2 $\mu$m CMOS vector A/D converter [10] and a 16 × 16 300 kHz 0.7 mW 2 $\mu$m CMOS analogue VQ [1]. These correspond to an energy per match

per dimension of 24 pJ and 9 pJ respectively. The integrated circuit (IC) described in this paper is distinguished from these approaches in that it is specifically targeted for the low power, low bandwidth application of ICEG classification and compression. Our chip achieves vector matching (without the winner take all function) to 7 bit 30 dimensional vectors with three coefficient linear prediction, at an energy consumption of 15 pJ per template per dimension using a 1.2 $\mu$m CMOS process. Although this figure is greater than that for [1] it should be noted that in [1] the mean absolute error metric is used rather than the squared Euclidean distance and no provision is provided for linear transformation of the incoming analogue vector.

## 2   ADAPTIVE DATA COMPRESSION

Recording of ICEGs in ICDs is currently very limited due to the amount of memory available and the power/area cost of implementing all but the simplest compression techniques. Micropower template matching however, enables large amounts of the signal to be encoded as template indices plus amplitude parameters. Effective compression of the ICEG requires adaptation to the short term non-stationary behaviour of the ICEG [2]. In particular, short term amplitude variations, lag variation, phase variation and ectopic beats (which originate from the ventricles of the heart and have differing morphology) reduce the achievable compression. The impact of ectopic beats can be reduced by increasing the number of templates. This can often be achieved without increasing the code book search complexity by using associated timing features. The amplitude and shift variations require short term adaptation of the template matching in order to minimise the residual error and hence raise the compression ratio at fixed distortion.

### 2.1   Amplitude and Shift Invariant Matching

In order to facilitate analogue implementation, a backward prediction procedure is used rather than the usual forward prediction [8]. This approach allows the incoming analogue template to be manipulated in the analogue domain for amplitude and shift invariance purposes. Consider the long term backward prediction problem described by,

$$r_b(n) = \tilde{x}(n) - b_0 x(n + \alpha) - b_1 \frac{\{x(n + \alpha + 1) - x(n + \alpha - 1)\}}{2} \tag{1}$$

where $r_b(n)$ denotes the backward residuals, $\tilde{x}$ is a template which is a function of previous beats, $x(\alpha)$ is the sampled ICEG signal, $\alpha$ the time index, $n$ is the template index and $b_0$ and $b_1$ are the amplitude and phase coefficients respectively. $b_0$ scales the current beat to match the template and hence is an amplitude term. $b_1$ scales the central difference of the current beat and is a function of the amplitude and phase corrections required to minimise the residuals. To see why this is a phase term consider the Taylor expansion of $Ax(t + \phi)$ to the first derivative term around t,

$$Ax(t + \phi) = Ax(t) + A\phi x'(t) \tag{2}$$

where $\phi$ is a small phase shift of $x(t)$ and $A$ is the amplitude factor. When $\phi$ is due to sampling jitter then, $-\frac{T}{2} \leq \phi \leq \frac{T}{2}$, where $T$ is the sampling period. Provided that $x(t)$ is sampled according to the Nyquist criterion, $\phi$ is sufficiently small for the first derivative term to adequately account for the sampling jitter. Hence, $b1$ accounts for the residual error remaining after optimisation of the integer $\alpha$. $\alpha$ is approximately determined by the beat detector of the ICD which attempts to detect the fiducial point of heart beats using filters and comparators. $b_0$ and $b_1$ can be determined by minimising the squared error between the current signal window and the previously recorded template which in this case has a closed form solution in terms of correlation coefficients. However, in Section 3 we present an alternative iterative procedure suited to low-power analogue implementations.

# 3 SYSTEM ARCHITECTURE & IMPLEMENTATION

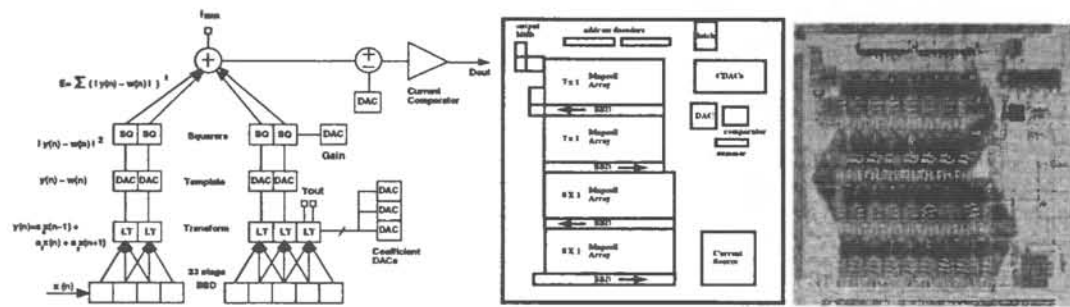

Figure 1: Left: Block diagram of the adaptive linear transform VQ chip. Middle: Floorplan of the chip. Right: Photomicrograph of the chip.

The ICEG is first high pass filtered to remove the DC and then is bandpass filtered to prevent aliasing and enhance the high frequency component for beat detection. (This is the filtering approach already existing in an ICD and therefore not implemented by us). This then feeds the discrete time analogue delay line, which is continuously sampling the signal at 250 Hz. The analogue samples are then transformed by a two layer network. The first layer implements the linear prediction by adjusting the amplitude $b_0$ and the phase of the analogue vector. Note that the phase consists of two components, the coarse part $\alpha$ corresponding to sample lags and the fine part $b_1$ corresponding to intra-sample lags. The second layer calculates the distance between the linearly predicted vector and the template $w(n)$ to be matched. A comparator is provided so that a match to within a given threshold may be detected.

## 3.1 Chip Architecture

Input to the IC is via a single analogue channel which is sampled by a bucket brigade device of length 30. The resultant 30 dimensional analogue vector is adaptively linear transformed to facilitate a shift and scale invariant match to a digital (7 bit per dimension) template. The IC generates digital representations of the square of the Euclidean distance between the transformed analogue vector and the digital template. A block diagram of the IC appears in Figure 1. The IC has been fabricated. Performance figures in this paper are based on measurements of the chip fabricated in a $1.2\mu m$ CMOS MOSIS process.

The block diagram shows the input signal being sampled by the bucket brigade device (BBD)[4]. The signal is sampled at a rate of 250 Hz. Existing circuitry in the defibrillator detects the peak of the heart beat and hence indicates a coarse alignment (due to detection jitter) to the template stored in the template DACs (TDACs). The BBD continues to sample until the coarse alignment is attained at which point the IC is biased up. The BBD now contains a segment of the ICEG corresponding to one heart beat. The digital error output is then monitored with the linear transform blocks configured to 1:1 mappings until an error minimum is detected indicating optimal sampling alignment. The three linear transform coefficient DACs (CDACs) which are common to the 30 linear transform blocks may then be adapted to further reduce the matching error. The transformation can be represented by $y(n) = a_0 x(n-1) + a_1 x(n) + a_2 x(n+1)$ where $a_0$ corresponds to CDAC0 etc. This constitutes a general linear long term prediction [8]. Constraining CDAC0 and CDAC2 to be equal magnitudes and opposite signs results in a minimisation of errors due to phase and amplitude variation and a simpler adaptation procedure. The matching error is computed via the squarer blocks and the summing node. The matching error consists of both a magnitude and exponent thereby increasing the dynamic range of the error representation.

The magnitude is the output of the squarer block. The exponent is determined by control of a current reference in the squaring circuit. A reference DAC and precision current comparator provide the means of successive approximation A/D conversion of the matching error current $I_{ERR}$. Using this scheme heart beat morphology can be classified by loading different templates (TDAC values). A stream of beats may be compressed by identifying matches with continuously updated representations of previous beats. Close matches are encoded by an index and an amplitude coefficent while poor matches are encoded by quantised residuals which have been minimised by the linear prediction.

### 3.2 Adaptation and Learning

The first step in the learning process is to determine $\alpha$, the coarse phase lag. This can be achieved by shifting the delay line and evaluating the error until a minimum is reached. Once the coarse phase lag $\alpha$ has been determined the error function to be minimised to compensate for amplitude and phase variations is given by $E = \sum_{i=1}^{N}(b_0 x_i + b_1 \Delta x_i - w_i)^2$, where the subscript $i$ implicitly incorporates the coarse phase $\alpha$. This is a quadratic in $b_0$ and $b_1$. $b_0$ and $b_1$ can be optimised separately provided cross terms in $E$ are negligible. Here the cross terms are given by $\sum_{i=1}^{N} 2 b_0 b_1 x_i \Delta x_i = b_0 b_1 (x_{N+1} x_N - x_1 x_0)$. Thus, if the end points of the $N$ point window have approximately the same value (as is usually the case for ICEG beats) then the cross terms in $E$ are negligible and $b_0$ and $b_1$ can be optimised separately.

So the only remaining issue is how to optimise a single parameter. A simple linear search takes at most $2^b$ evaluations of $E$ where $b$ is the number of bits. A search based on bisection takes $b + 2$ evaluations. Techniques involving gradient descent and conjugate gradient lead to more complex learning logic with minor reductions in the number of evaluations. Therefore, bisection is the best compromise between the number of evaluations and the complexity of the learning state machine.

Once the best template match has been achieved, learning may also then be applied to the template itself depending on the application and context. For example, in the case of adaptive classification a weight perturbation algorithm [6] could be used to adapt the template for morphological drift based on heart rate information. Similarly, for a data compression application, if the template match exceeds a fidelity criterion the template may be adapted and the template changes logged in the compression record.

### 3.3 Building Blocks

In order to implement the template matcher, sub-threshold analogue VLSI building blocks were designed. All transistors in the building blocks operate in weak inversion exclusively. We do not have the space to describe all of the building blocks, so we will focus here on the linear transform and squarer cells.

### 3.3.1 Linear Transform Cell

The linear transform (LT) cell consists of three linearised differential pairs [7] with their biases controlled by the coefficient DACs (CDACs) (see Figure 2(a)). The nature of the linearisation is controlled by the ratio of the aspect ratios of M3 to M5 and M4 to M6. Methods for choosing this ratio are discussed in [5]. Denoting the aspect ratio of a transistor by $S$ we chose $S_3/S_5 = S_4/S_6 = 4$. This introduces some ripple in the transconductance while increasing the asymptotic saturation voltage to $4nU_T$ compared to $nU_T$ for the ordinary differential pair. Signed coefficients are achieved by switches at the outputs of the differential pairs. The template DACs (TDACs) have differential outputs to form the difference $y(n) - w(n)$ where $w(n)$ is the $n$th template value.

### 3.3.2   Squaring Cell

The squaring function must meet the following design constraints. It should have current inputs and outputs in order to avoid linear current to voltage conversion at low currents. The squared current must be normalised to the original linear range to avoid excessive power consumption. The squaring function should avoid the MOS square law approach in order to conserve space and power, and the the available voltage range should be 3.3 V rail to rail.

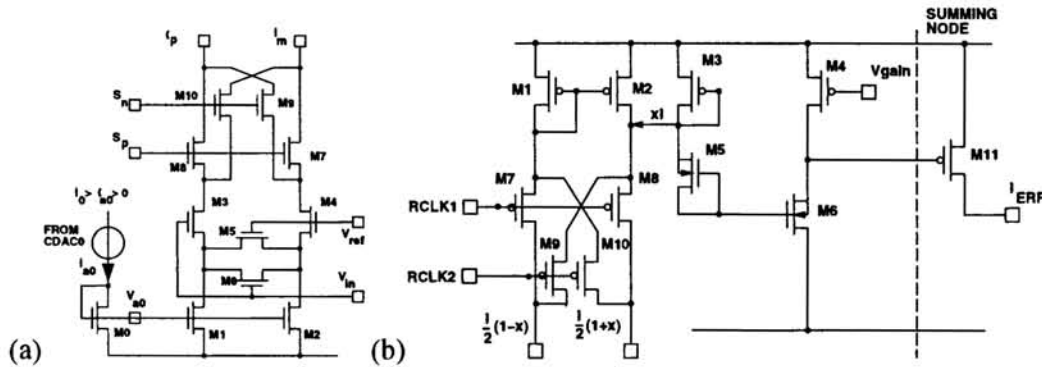

Figure 2: (a) Circuit diagram of one of three the linear transform linearised differential pairs in the LT cell. (b) Circuit diagram of the squarer (SQ cell) and the summing node.

The choices available then are restricted to weak inversion circuits. The circuit (see Figure 2(b)) used relies on the translinear principle [9]. Here, loops of MOS g-s diode structures operating in weak inversion are used to form a normalised squared current which is summed to form the final normalised output. The translinear loops are implemented with P-type transistors in separate N-wells to avoid the body effect. Positive and negative inputs are squared separately using the RCLK signals and then added at the output.

### 3.4   Circuit Performance

Table 1: Summary of electrical specifications of the chip.

| Item | Conditions | Value |
|---|---|---|
| Template dimension | | 30 |
| Adaptation coefficients | Excludes squarer error gain control | 3 |
| DAC Precision | Weighted lateral PNP | 7 bits |
| Max. Error per dimension[a] | CDACx=64, DCBBD, w/r to TDACs | 2 bits |
| LSB bias | | 2 nA |
| Power comsumption | TDACs=CDAC1=64, duty cycle[b] = 3.2% | 2.5 $\mu$W |

[a] Excludes error at 1st CDAC0 stage. [b] For 1 bpm, chip biased up 8/250 of the time.

We provide three measures of the performance of the chip along with a summary of its basic electrical characteristics which is shown in Table 1. The first measure characterises the accuracy of the template matching function relative to the available precision of the template. This is summarised by the Maximum Error per dimension in Table 1 which was produced by inputing a zero offset DC signal into the BBD and setting each CDAC in turn to one half of its maximum value. The TDACs were then adjusted so as to minimise the output of the squarer. Therefore, the resulting TDAC values indicate the accumulated effects of transistor mismatches through each path to the squarer output. The curves generated are averages over 80 trials to remove noise influences (where as the classification performance

shown in Table /refvterr-tab includes such influences). The curves showed that except for the input stage corresponding to CDAC0 (stage 30) the accumulated mismatches influence the two least significant bits of the TDACs. A larger error of 4 bits for the first stage feeding CDAC0 was due to a design oversight of not providing a dummy capacitive load to the input end of the BBD (stage 30 of CDAC0 derives its input from the input BBD cell, which does not have the full capacitive loading of three linearised differential pairs as on the rest of the cells).

Table 2: Relative impact on the error output of the chip for the adaptation steps of alignment, amplitude and phase correction for patient No. 2s ST rhythm. The errors are normalised to the non-aligned error. A numerical simulation is provided for comparison to the chip performance.

| Adaptation step | Chip Error | Std. Dev. | Simulation Error | Std. Dev. |
|---|---|---|---|---|
| No align | 1.0 | 0.04 | 1.0 | 0.28 |
| Align | 0.31 | 0.07 | 0.41 | 0.35 |
| Amplitude | 0.16 | 0.05 | 0.37 | 0.22 |
| Phase | 0.07 | 0.01 | 0.32 | 0.16 |

The second performance measure uses real heart patients ICEG (Sinus Tachycardia) ST data. Table 2 shows the normalised output error of the chip averaged over 107 heart beats while being compared to the 10th beat in the series. The normalised error was measured from a mirrored version of the current at the output of the chip. The adaptation steps shown in the table are as follows. "No align" implies that the error for the template match is determined only by the approximate alignment provided by a numerical simulation of the beat detector of the ICD. "Align" corresponds to coarse alignment where the matching error is calculated up to two samples either side to determine the best positioning of the input in the BBD. "Amplitude" corresponds to adaptation of the amplitude coefficient by adjustment of CDAC1. "Phase" corresponds to adaptation of the difference between CDAC2 and CDAC0. Each of the adaptations reduces the error of the match with the coarse alignment being most significant. An idealised limited precision numerical simulation of the error calculation is also provided in the table for comparison. It can be seen that the amplitude and phase adaptation steps lower the relative error more for the chip than in the simulation. This is most likely due to the adaptation on the chip also compensating for the analogue noise and imprecision as well as the variability of the original data.

The third performance measure illustrates the ability of the chip to solve a blind classification problem and is summarised in Table 3. The safe rhythm of the patient is Sinus Tachycardia (ST). For each patient one beat is chosen at random as the template and is loaded into the TDACs of the chip. The 20 beats subsequent to the chosen template are then used to determine the average error between templates after adaptation. Twice this error is then used as the classifier threshold for "safe" versus "unknown". The ST and VT data sets for the patient are then passed through the chip and classified giving the column "% Correct chip". For comparison the expected best performance for the data set are also reproduced in the table from previous work by the authors [3]. The results indicate that a very simple blind classification algorithm when combined with the adaptive template matching capabilities of the chip shows good performance for 4 out of 5 patients.

## 4  CONCLUSION

We have presented a micropower learning vector quantization system that can provide hardware support for both signal classification and compression of ICEG signals. The analogue block can be used to implement several different classification and compression algorithms

Table 3: Performance of the chip on a blind classification task for 5 patients with Ventricular Tachycardia (VT) 1:1 retrograde conduction compared to classification bounds.

| Patient | No. of Complexes | | % Correct chip | | % Correct Bounds | |
|---------|------|------|------|------|------|------|
|         | ST   | VT   | ST   | VT   | ST   | VT   |
| 1[a]    | 440  | 61   | 99   | 100  | 99   | 100  |
| 2       | 107  | 71   | 99   | 100  | 99   | 100  |
| 3       | 177  | 75   | 89   | 100  | 98   | 100  |
| 4       | 110  | 71   | 92   | 100  | 99   | 100  |
| 5       | 38   | 90   | 97   | 87   | 99   | 100  |

[a] The R point search interval was increased to 4 for this patient.

depending on how the template matching capability is utilised. By providing significant compression capability in an ICD, a larger data base of natural onset cardiac arrhythmia should become available, leading to improved designs of ICD based adaptive classification and compression systems.

## 5 ACKNOWLEDGEMENTS

The work in this paper was funded by the Australian Research Council and Telectronics Pacing Systems Ltd, Sydney, Australia.

## References

[1] G. Cauwenberghs and V. Pedroni. A Charge-Based CMOS Parallel Analog Vector Quantiser. In *NIPS*, volume 7, pages 779–786. MIT Press, 1995.

[2] R.J. Coggins. *Low Power Signal Compression and Classification for Implantable Defibrillators*. PhD thesis, University of Sydney, Sydney, Australia, 1996.

[3] R.J. Coggins and M.A. Jabri. Classification and Compression of ICEGs using Gaussian Mixture Models. In J. Principe, L. Giles, N. Morgan, and E. Wilson, editors, *Neural Networks for Signal Processing*, volume 7, pages 226–235. IEEE, 1997.

[4] R.J. Coggins, M.A. Jabri, B.G. Flower, and S.J. Pickard. A Hybrid Analog and Digital VLSI Neural Network for Intracardiac Morphology Classification. *IEEE Journal of Solid-State Circuits*, 30(5):542–550, May 1995.

[5] M. Furth and A. Andreou. Linearised Differential Transconductors in Subthreshold CMOS. *Electronics Letters*, 31(7):545–547, 1995.

[6] M.A. Jabri and B.G. Flower. Weight Perturbation: An Optimal Architecture and Learning Technique for Analog VLSI Feedforward and Recurrent Multilayer Networks. *IEEE Transactions on Neural Networks*, 3(1):154–157, January 1992.

[7] F. Krummenacher and N Joehl. A 4Mhz CMOS Continuous Time Filter with On Chip Automatic Tuning. *IEEE Journal of Solid-State Circuits*, 23(3):750–758, June 1986.

[8] G. Nave and A. Cohen. ECG Compression Using Long Term Prediction. *IEEE Trans. Biomed. Eng.*, 40(9):877–885, 1993.

[9] E. Seevinck. *Analysis and Synthesis of Translinear Integrated Circuits*. Elsevier, 1988.

[10] G.T. Tyson, S. Fallahi, and A.A. Abidi. An 8b CMOS Vector A/D converter. In *Proceedings of the International Solid State Circuits Conference*, pages 38–39, 1993.
